# Adaptive Retina with Center-Surround Receptive Field

**Shih-Chii Liu and Kwabena Boahen**
Computation and Neural Systems
139-74 California Institute of Technology
Pasadena, CA 91125
shih@pcmp.caltech.edu, buster@pcmp.caltech.edu

## Abstract

Both vertebrate and invertebrate retinas are highly efficient in extracting contrast independent of the background intensity over five or more decades. This efficiency has been rendered possible by the adaptation of the DC operating point to the background intensity while maintaining high gain transient responses. The center-surround properties of the retina allows the system to extract information at the edges in the image. This silicon retina models the adaptation properties of the receptors and the antagonistic center-surround properties of the laminar cells of the invertebrate retina and the outer-plexiform layer of the vertebrate retina. We also illustrate the spatio-temporal responses of the silicon retina on moving bars. The chip has 59x64 pixels on a 6.9x6.8mm$^2$ die and it is fabricated in 2 $\mu m$ n-well technology.

## 1 Introduction

It has been observed previously that the initial layers of the vertebrate and invertebrate retina systems perform very similar processing functions on the incoming input signal[1]. The response versus log intensity curves of the receptors in invertebrate and vertebrate retinas look similar. The curves show that the receptors have a larger gain for changes in illumination than to steady illumination, i.e, the receptors adapt. This adaptation property allows the receptor to respond over a large input range without saturating.

Anatomically, the eyes of invertebrates differ greatly from that of vertebrates. Ver-

tebrates normally have two simple eyes while insects have compound eyes. Each compound eye in the fly consists of 3000-4000 ommatidia and each ommatidium consists of 8 photoreceptors. Six of these receptors (which are also called R1-R6) are in a single spectral class. The other two receptors, R7 and R8 provide channels for wavelength discrimination and polarization.

The vertebrate eye is divided into the outer-plexiform layer and the inner-plexiform layer. The outer-plexiform layer consists of the rods and cones, horizontal cells and bipolar cells. Invertebrate receptors depolarise in response to an increase in light, in contrast to vertebrate receptors, which hyperpolarise to an increase in light intensity. Both vertebrate and invertebrate receptors show light adaptation over at least five decades of background illumination. This adaptation property allows the retina to maintain a high transient gain to contrast over a wide range of background intensities.

The invertebrate receptors project to the next layer which is called the lamina layer. This layer consists primarily of monopolar cells which show a similar response versus log intensity curve to that of vertebrate bipolar cells in the outer-plexiform layer. Both cells respond with graded potentials to changes in illumination. These cells also show a high transient gain to changes in illumination while ignoring the background intensity and they possess center-surround receptive fields. In vertebrates, the cones which are excited by the incoming light, activate the horizontal cells which in turn inhibit the cones. The horizontal cells thus mediate the lateral inhibition which produces the center-surround properties. In insects, a possible process of this lateral inhibition is done by current flow from the photoreceptors through the epithelial glial cells surrounding an ommatidium or the modulation of the local field potential in the lamina to influence the transmembrane potential of the photoreceptor[2]. The center-surround receptive fields allow contrasts to be accentuated since the surround computes a local mean and subtracts that from the center signal.

Mahowald[3] previously described a silicon retina with adaptive photoreceptors and Boahen et al.[4] recently described a compact current-mode analog model of the outer-plexiform layer of the vertebrate retina and analysed the spatio-temporal processing properties of this retina[5]. A recent array of photoreceptors from Delbrück[6] uses an adaptive photoreceptor circuit that adapts its operating point to the background intensity so that the pixel shows a high transient gain over 5 decades of background illumination. However this retina does not have spatial coupling between pixels.

The pixels in the silicon retina described here has a compact circuit that incorporates both spatial and temporal filtering with light adaptation over 5 decades of background intensity. The network exhibits center-surround behavior. Boahen *et al.*[4] in their current-mode diffusor retina, draw an analogy between parts of the diffusor circuit and the different cells in the outer-plexiform layer. While the same analogy cannot be drawn from this silicon retina to the invertebrate retina since the function of the cells are not completely understood, the output responses of the retina circuit are similar to the output responses of the photoreceptor and monopolar cells in invertebrates.

The circuit details are described in Section 2 and the spatio-temporal processing performed by the retina on stimulus moving at different speeds is shown in Section

3.

## 2 Circuit

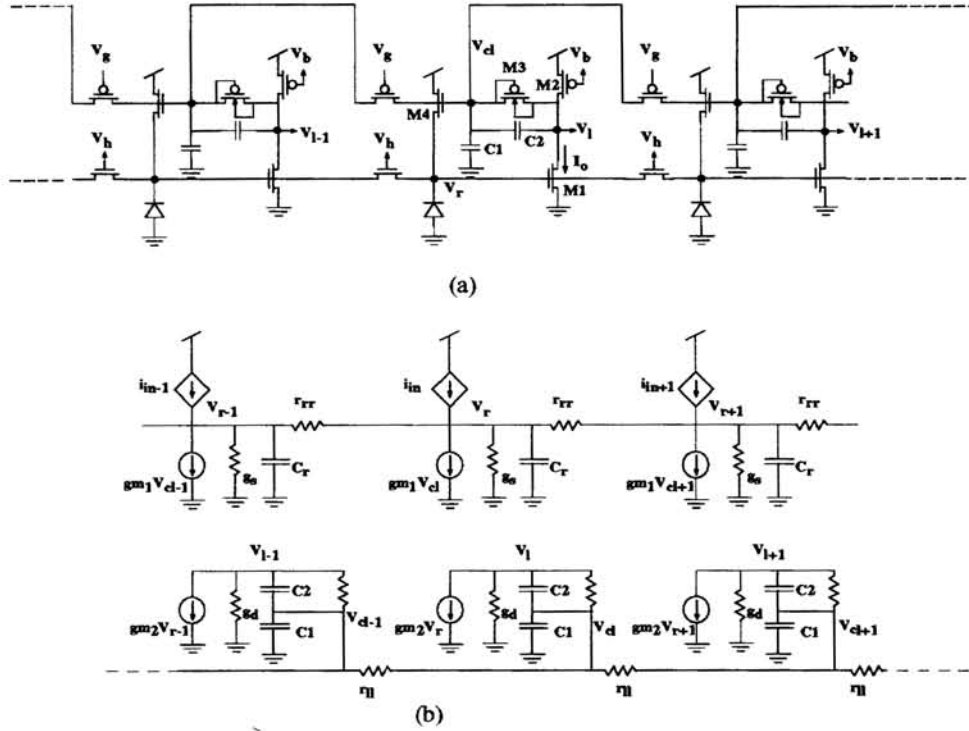

(a)

(b)

Figure 1: (a) One-dimensional version of the retina. (b) Small-signal equivalent of circuit in (a).

A one-dimensional version of the retina is shown in Figure 1(a). The retina consists of an adaptive photoreceptor circuit at each pixel coupled together with diffusors, controlled by voltages, $V_g$ and $V_h$. The output of this network can either be obtained at the voltage output, $V_l$ or at the current output, $I_o$ but the outputs have different properties. Phototransduction is obtained by using a reverse-biased photodiode which produces current that is proportional to the incident light. The logarithmic properties are obtained by operating the feedback transistor shown in Figure 1(a) in the subthreshold region. The voltage change at the output photoreceptor, $v_r$, is proportional to a small contrast since

$$v_r = \frac{U_T}{\kappa} d(log I) = \frac{U_T}{\kappa} \frac{dI}{I} = \frac{U_T}{\kappa} \frac{i}{I_{bg}}$$

where $U_T$ is the thermal voltage, $\kappa = \frac{C_{ox}}{C_{ox}+C_d}$ , $C_{ox}$ is the oxide capacitance and $C_d$ is the depletion capacitance of a transistor. The circuit works as follows: If the photocurrent through the photodiode increases, $V_r$ will be pulled low and the output voltage at $V_l$ increases by $v_l = Av_r$ where $A$ is the amplifier gain of the output stage. This output change in $V_l$ is coupled into $V_{cl}$ through a capacitor

divider ratio, $\frac{C_2}{C_1+C_2}$. The feedback transistor, M4, operates in the subthreshold region and supplies the current necessary to offset the photocurrent. The increase in $V_{cl}$ (i.e. the gate voltage of M4) causes the current supplied by M3 to increase which pulls the node voltage, $V_r$, back to the voltage level needed by M1 to sink the bias current from transistor, M2.

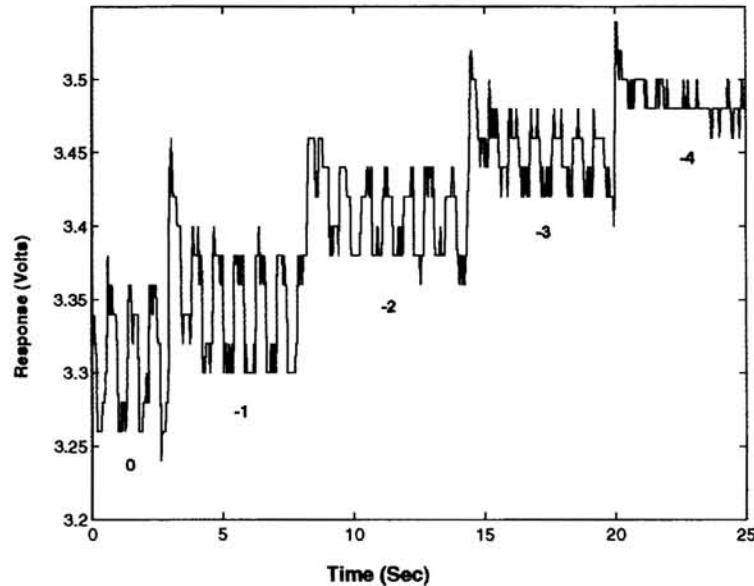

Figure 2: This figure shows the output response of the receptor to a variation of about 40% p-p in the intensity of a flickering LED light incident on the chip. The response shows that the high sensitivity of the receptor to the LED is maintained over 5 decades of differing background intensities. The numbers on the section of the curve indicate the log intensity of the mean value. 0 log is the absolute intensity from the LED.

The adaptive element, M3, has an I-V curve which looks like a hyperbolic sine. The small slope of the I-V curve in the middle means that for small changes of voltages across M3, the element looks like an open-circuit. With large changes of voltage across M3, the current through M3 becomes exponential and $V_{cl}$ is charged or discharged almost instantaneously.

Figure 2 shows the output response of the photoreceptor to a square-wave variation of about 40% p-p in the intensity of a red LED (635 nm). The results show that the circuit is able to discern the small contrast over five decades of background intensity while the steady-state voltage of the photoreceptor output varies only about $15mV$. Further details of the photoreceptor circuit and its adaptation properties are described in Delbrück[6].

## 3  Spatio-Temporal Response

The spatio-temporal response of the network to different moving stimuli is explored in this section. The circuit shown in Figure 1(a) can be transferred to an equivalent network of resistors and capacitors as shown in Figure 1(b) to obtain the transfer function of the circuit. The capacitors at each node are necessary to model the

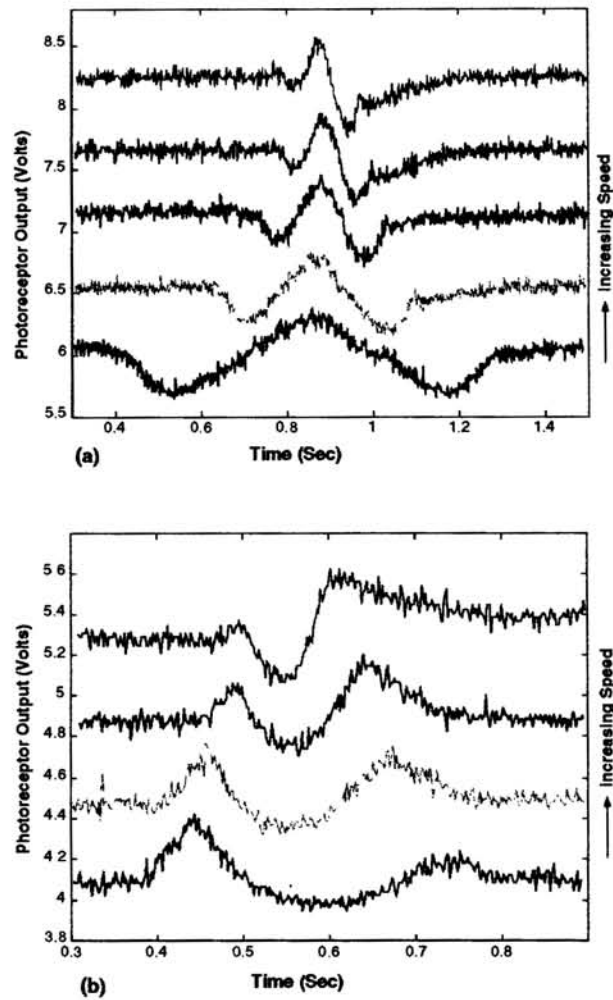

Figure 3: (a) Response of a pixel to a grey strip 2 pixels wide of gray-level "0.4" on a dark background of level "0" moving past the pixel at different speeds. (b) Response of a pixel to a dark strip of gray-level "0.6" on a white background of level "1" moving past the pixel at different speeds. The voltage shown on these curves is not the direct measurement of the voltage at $V_l$ but rather $V_l$ drives a current-sensing transistor and this current is then sensed by an offchip current sense-amplifier.

temporal responses of the circuit.

The chip results from the experiments below illustrate the center-surround proper-
ties of the network and the difference in time-constants between the surround and
center.

## 3.1  Chip Results

Data from the 2D chip is shown in the next few figures. In these experiments, we
are only looking at one pixel of the 2D array. A rotating circular fly-wheel stimulus
with strips of alternating contrasts is mounted above the chip. The stimulus was
created using Mathematica. Figure 3a shows the spatio-temporal impulse response
of one pixel measured at $V_l$ with a small strip at level "0.4" on a dark background of
level "0" moving past the pixels on the row. At slow speeds, the impulse response
shows a center-surround behavior where the pixel first receives inhibition from the
preceding pixels which are excited by the stimulus. When the stimulus moves by
the pixel of interest, it is excited and then it is inhibited by the subsequent pixels
seeing the stimulus.

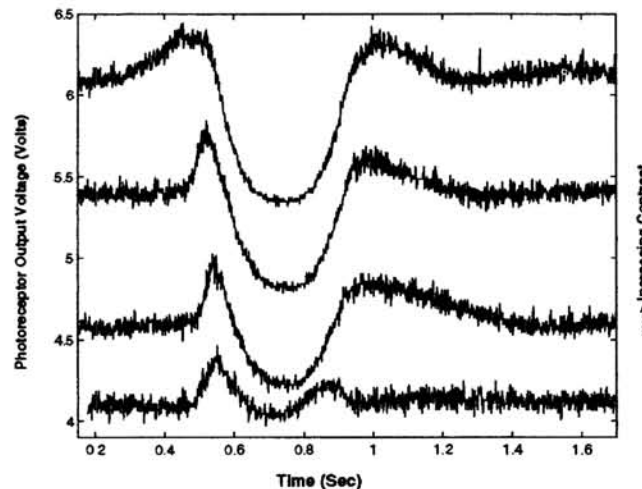

Figure 4: Response of a pixel to a strip of varying contrasts on a dark background
moving past the pixel at a constant speed.

At faster speeds, the initial inhibition in the response grows smaller until at some
even faster speed, the initial inhibition is no longer observed. This response comes
about because the inhibition from the surround has a longer-time constant than the
center. When the stimulus moves past the pixel of interest, the inhibition from the
preceding pixels excited by the stimulus does not have time to inhibit the pixel of
interest. Hence the excitation is seen first and then the inhibition comes into place
when the stimulus passes by. Note that in these figures (Figures 3-4), the curves
have been displaced to show the pixel response at different speeds of the moving
stimulus. The voltage shown on these curves is not the direct measurement of the
voltage at $V_l$ but rather $V_l$ drives a current-sensing transistor and this current is
then sensed by an off-chip current sense-amplifier.

Figure 3b shows the spatio-temporal impulse response of one pixel with a similar

size strip of level "0.6" on a light background of level "1" moving past the row of pixels. The same inhibition behavior is seen for increasing stimulus speeds. Figure 4 shows the output response at $V_l$ for the same stimulus of gray-levels varying from "0.2" to "0.8" on a dark background of level "0" moving at one speed. The peak excitation response is plotted against the contrast in Figure 5. A level of "0.2" corresponds to a irradiance of $15mW/m^2$ while a level of "0.8" corresponds to a irradiance of $37.4mW/m^2$. These measurements are done with a photometer mounted about 1.5in above a piece of paper with the contrast which is being measured. The irradiance varies exponentially with increasing level.

## 4   Conclusion

In this paper, we described an adaptive retina with a center-surround receptive field. The system properties of this retina allows it to model functionally either the responses of the laminar cells in the invertebrate retina or the outer-plexiform layer of vertebrate retina. We show that the circuit shows adaptation to changes over 5 decades of background intensities. The center-surround property of the network can be seen from its spatio-temporal response to different stimulus speeds. This property serves to remove redundancy in space and time of the input signal.

**Acknowledgements**

We thank Carver Mead for his support and encouragement. SC Liu is supported by an NIMH fellowship and K Boahen is supported by a Sloan fellowship. We thank Tobias Delbrück for the inspiration and help in testing the design. We also thank Rahul Sarpeshkar and Bradley Minch for comments. Fabrication was provided by MOSIS.

## References

[1] S. B. Laughlin, "Coding efficiency and design in retinal processing", In: *Facets of Vision* (D. G. Stavenga and R. C. Hardie, eds) pp. 213-234. Springer, Berlin, 1989.

[2] S. R. Shaw, "Retinal resistance barriers and electrical lateral inhibition", *Nature, Lond.***255**,: 480-483, 1975.

[3] M. A. Mahowald, "Silicon Retina with Adaptive Photoreceptors" in *SPIE/SPSE Symposium on Electronic Science and Technology: From Neurons to Chips.* Orlando, FL, April 1991.

[4] K. A. Boahen and A. G. Andreou, "A Contrast Sensitive Silicon Retina with Reciprocal Synapses", In D. S. Touretzky (ed.), *Advances in Neural Information Processing Systems 4*, 764-772. San Mateo, CA: Morgan Kaufmann, 1992.

[5] K. A. Boahen, "Spatiotemporal sensitivity of the retina: A physical model", *CNS Memo CNS-TR-91-06*, California Institute of Technology, Pasadena, CA 91125, June 1991.

[6] T. Delbrück, "Analog VLSI Phototransduction by continous-time, adaptive, logarithmic photoreceptor circuits", *CNS Memo No.30*, California Institute of Technology, Pasadena, CA 91125, 1994.